# Synergy Of Clustering Multiple Back Propagation Networks

**William P. Lincoln***  **and Josef Skrzypek†**
UCLA Machine Perception Laboratory
Computer Science Department
Los Angeles, CA 90024

## ABSTRACT

The properties of a cluster of multiple back-propagation (BP) networks are examined and compared to the performance of a single BP network. The underlying idea is that a synergistic effect within the cluster improves the performance and fault tolerance. Five networks were initially trained to perform the same input-output mapping. Following training, a cluster was created by computing an average of the outputs generated by the individual networks. The output of the cluster can be used as the desired output during training by feeding it back to the individual networks. In comparison to a single BP network, a cluster of multiple BP's generalization and significant fault tolerance. It appear that cluster advantage follows from simple maxim "you can fool some of the single BP's in a cluster all of the time but you cannot fool all of them all of the time" {Lincoln}

## 1  INTRODUCTION

Shortcomings of back-propagation (BP) in supervised learning has been well documented in the past {Soulie, 1987; Bernasconi, 1987}. Often, a network of a finite size does not learn a particular mapping completely or it generalizes poorly. Increasing the size and number of hidden layers most often does not lead to any improvements {Soulie,

1987}. The central question that this paper addresses is whether a "synergy" of clustering multiple back-prop nets improves the properties of the clustered system over a comparably complex non-clustered system. We use the formulation of back-prop given in {Rumelhart, 1986}. A cluster is shown in figure 1. We start with five, three-layered, back propagation networks that "learn" to perform the same input-output mapping. Initially the nets are given different starting weights. Thus after learning, the individual nets are expected to have different internal representations. An input to the cluster is routed to each of the nets. Each net computes its output and the judge uses these outputs, $\hat{y}_k$ to form the cluster output, $\hat{y}$. There are many ways of forming $\hat{y}$ but for the sake of simplicity, in this paper we consider the following two rules:

$$simple \;\; average : \hat{y} = \sum_{K=1}^{N} \frac{1}{N} \hat{y}_k \qquad (1.1)$$

$$convex \;\; combination : \hat{y} = \sum_{K=1}^{N} W_k y_k \qquad (1.2)$$

Cluster function 1.2 adds an extra level of fault tolerance by giving the judge the ability to bias the outputs based on the past reliability of the nets. The $W_k$ are adjusted to take into account the recent reliability of the net. One weight adjustment rule is $W_k = W_k \cdot G \cdot \frac{e}{e_k}$ where $e = \frac{1}{N} \sum_{k=1}^{N} e_k$, G is the gain of adjustment and $e_k = ||\hat{y} - \hat{y}_k||$ is the network deviation from the cluster output. Also, in the absence of an initial training period with a perfect teacher the cluster can collectively self-organize. The cluster in this case is performing an "averaging" of the mappings that the individual networks perform based on their initial distribution of weights. Simulations have been done to verify that self organization does in fact occur. In all the simulations, convergence occurred before 1000 passes.

Besides improved learning and generalization our clustered network displays other desirable characteristics such as fault tolerance and self-organization. Feeding back the cluster's output to the N individual networks as the desired output in training endows the cluster with fault tolerance in the absence of a teacher. Feeding back also makes the cluster continuously adaptable to changing conditions. This aspects of clustering is similar to the tracking capabilities of adaptive equalizers. After the initial training period it is usually assumed that no teacher is present, or that a teacher is present only at relatively infrequent intervals. However, if the failure rate is large enough, the performance of a single, non-clustered net will degrade during the periods when no teacher is present.

## 2  CLUSTERING WITH FEEDBACK TO INCREASE FAULT TOLERANCE IN THE ABSENCE OF A PERFECT TEACHER.

When a teacher is not present, $\hat{y}$ can be used as the desired output and used to continuously train the individual nets. In general, the correct error that should be back-propagated, $d_k = y - \hat{y}_k$, will differ from the actual error, $\hat{d}_k = \hat{y} - \hat{y}_k$. If $d_k$ and $\hat{d}_k$ differ significantly, the error of the individual nets (and thus the cluster as a whole) can increase

over time. This phenomenon is called drift. Because of drift, retraining using $\hat{y}$ as the desired output may seem disadvantageous when no faults exist within the nets. The possibility of drift is decreased by training the nets to a sufficiently small error. In fact under these circumstance with sufficiently small error, it is possible to see the error to decrease even further.

It is when we assume that faults exist that retraining becomes more advantageous. If the failure rate of a network node is sufficiently low, the injured net can be retrained using the judge's output. By having many nets in the cluster the effect of the injured net's output on the cluster output can be minimized. Retraining using $\hat{y}$ adds fault tolerance but causes drift if the nets did not complete learning when the teacher was removed.

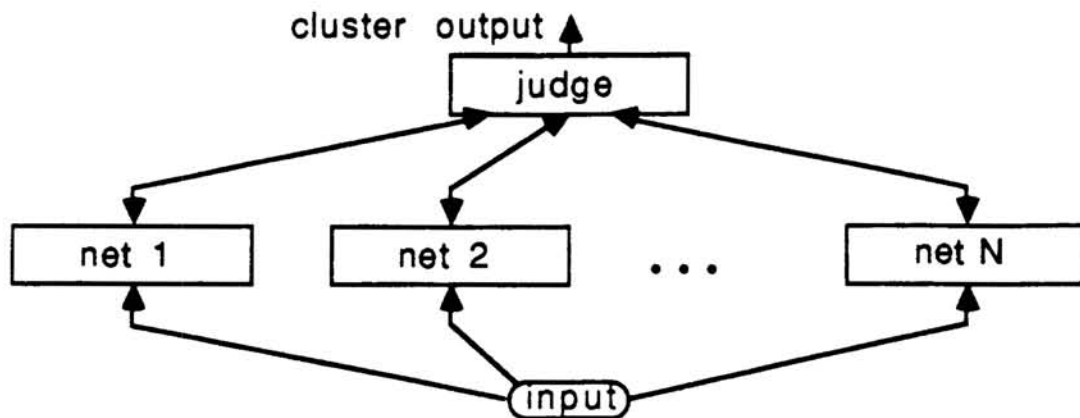

**Figure 1:** A cluster of N back-prop nets.

## 3   EXPERIMENTAL METHODS.

To test the ideas outlined in this paper an abstract learning problem was chosen. This abstract problem was used because many neural network problems require similar separation and classification of a group of topologically equivalent sets in the process of learning {Lippman, 1987}. For instance, images categorized according to their characteristics. The input is a 3-dimensional point, $P = (x,y,z)$. The problem is to categorize the point P into one of eight sets. The 8 sets are the 8 spheres of radius 1 centered at $x = (\pm 1)$, $y = (\pm,1)$, $z = (\pm,1)$ The input layer consists of three continuous nodes. The size of the output layer was 8, with each node trained to be an indicator function for its associated sphere. One hidden layer was used with full connectivity between layers. Five nets with the above specifications were used to form a cluster. Generalization was tested using points outside the spheres.

# 4   CLUSTER ADVANTAGE.

The performance of a single net is compared to performance of a five net cluster when the nets are not retrained using $\hat{y}$. The networks in the cluster have the same structure and size as the single network. Average errors of the two systems are compared. A useful measure of the cluster advantage is obtained by taking the ratio of an individual net's error to the cluster error. This ratio will be smaller or larger than 1 depending on the relative magnitudes of the cluster and individual net's errors. Figures 2a and 2b show the cluster advantage plotted versus individual net error for 256 and 1024 training passes respectively. It is seen that when the individual nets either learn the task completely or don't learn at all there is not a cluster advantage. However, when the task is learned even marginally, there is a cluster advantage.

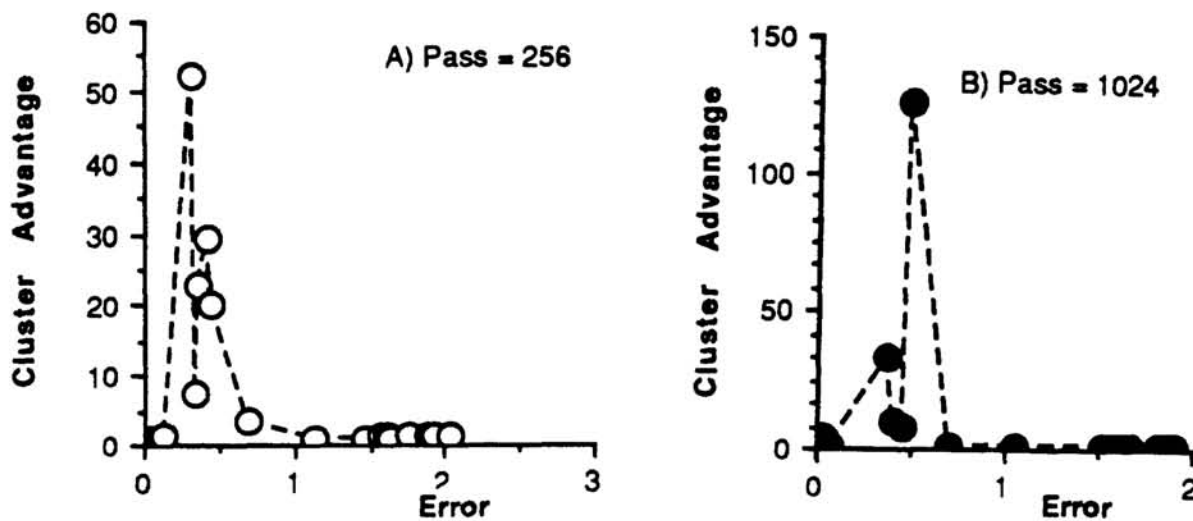

**Figure 2:** Cluster Advantage versus Error.
Data points from more than one learning task are shown.
A) After 256 training passes. B) After 1024 training passes.

The cluster's increased learning is based on the synergy between the individual networks and not on larger size of a cluster compared to an individual network. An individual net's error is dependent on the size of the hidden layer and the length of the training period. However, in general the error is not a decreasing function of the size of the hidden layer throughout its domain, i.e. increasing the size of the hidden layer does not always result in a decrease in the error. This may be due to the more direct credit assignment with the smaller number of nodes. Figures 4a and 4b show an individual net's error versus hidden layer size for different training passes. The point to this pedagogics is to counter the anticipated argument: "a cluster should have a lower error based on the fact that it has more nodes".

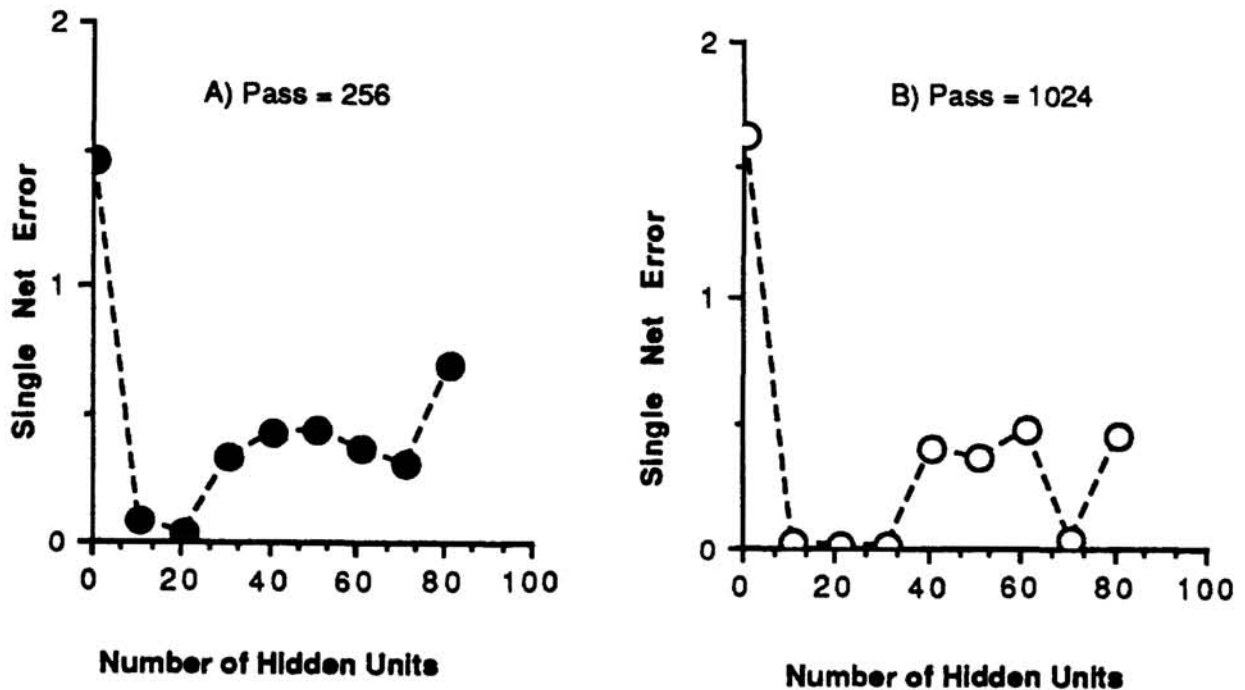

**Figure 3:** Error of a single BP network is a nonlinear funtion
of the number of hidden nodes.
A) After 256 training passes  B) After 1024 training passes

## 5  FAULT TOLERANCE.

the judge's output as the desired output and retraining the individual networks, fault tolerance is added. The fault tolerant capabilities of a cluster of 5 were studied. The size of the hidden layer is 15. After the nets were trained, a failure rate of 1 link in the cluster per 350 inputs was introduced. This failure rate in terms of a single unclustered net is 1 link per 1750 (=5.350) inputs. The link that is chosen to fail in the cluster was randomly selected from the links of all the networks in the cluster. When a link failed its weight was set to 0. The links from the nets to the judge are considered immune from faults in this comparison. A pass consisted of 1 presentation of a random point from each of the 8 spheres. Figure 4 shows the fault tolerant capabilities of a cluster. By knowing the behavior of the single net in the presence of faults, the fault tolerant behavior of any conventional configuration (i.e. comparison and spares) of single nets can be determined, so that this form of fault tolerance can be compared with conventional fault tolerant schemes.

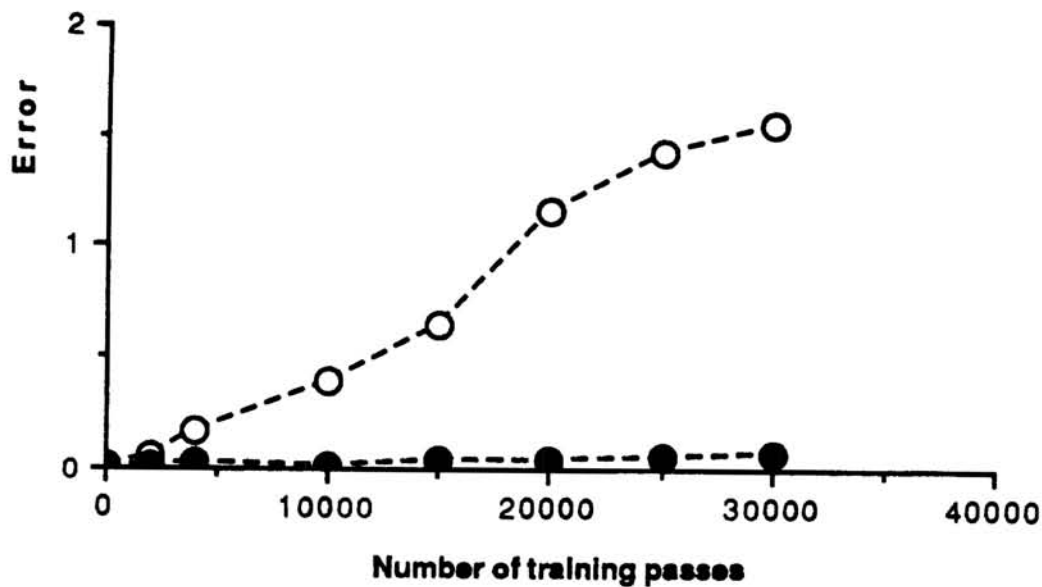

**Figure 4:** Fault tolerance of a cluster using feedback
from the "judge" as a desired training output.
Error as a function of time (# of training passes) without link
failures (solid circles) and with link failures (open cirles).
Link failure rate = 1 cluster link per 350 inputs
or 1 single net link per 1750 (=5 nets*350) inputs

## 6 CONCLUSIONS.

Clustering multiple back-prop nets has been shown to increase the performance and fault tolerance over a single network. Clustering has exhibited very interesting self organization. Preliminary investigations are restricted to a few simple examples. Nevertheless, there are some interesting results that appear to be rather general and which can thus be expected to remain valid for much larger and complex systems. The clustering ideas presented in this paper are not specific to back-prop but can apply to any nets trained with a supervised learning rule. The results of this paper can be viewed in an enlightening way. Given a set of weights, the cluster performs a mapping. There is empirical evidence of local minimum in this "mapping space". The initial point in the mapping space is taken to be when the cluster output begins to be fed back. Each time a new cluster output is fed back the point in the mapping space moves. The step size is related to the step size of the back prop algorithm. Each task is conjectured to have a local minimum in the mapping space. If the point moves away from the desired local minimum, drift occurs. A fault moves the point away from the local minimum. Feedback moves the point closer to the local minimum. Self organization can be viewed as finding the local minimum of the valley that the point is initially placed based on the initial distribution of weights.

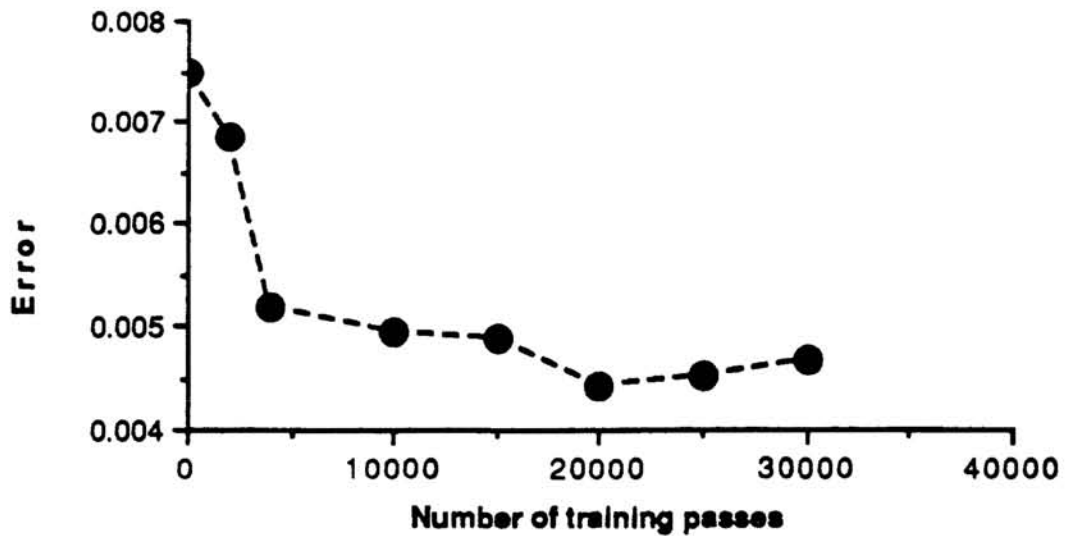

**Figure 5:** Cluster can continue to learn in the absence of a
teacher if the feedback from the judge is used as a
desired training output. No link failures.

## 6.1   INTERPRETATION OF RESULTS.

The results of the previous section can be interpreted from the viewpoint of the model
described in this section. This model attempts to describe how the state of the nets change
due to possibly incorrect error terms being back-propagated, and how in turn the state of
the net determines its performance. The state of a net could be defined by its weight
string. Given its weight string, there is a duality between the mapping that the net is per-
forming and its error. When a net is being trained towards a particular mapping, its
current weight string determines the error of the net. The back-propagation algorithm is
used to change the weight string so that the error decreases. The duality is that at any
time a net is performing some mapping (it may not be the desired mapping) it is perform-
ing that mapping with no error. This duality has significance in connection with self-
organization which can be viewed as taking an "average" of the N mappings.

While the state of a net could be defined by its weight string, a state transition due to a backward error propagation is not obvious. A more useful definition of the state of a net is its error. (The error can be estimated by taking a representative sample of input vectors and propagating them through the net and computing the average error of the outputs.) Having defined the state, a description of the state transition rules can now be given.

output of net (i) = f ( state of net (i) , input )

state of net (i) = g ( state of net (i) , output of net (1) ,...,output of net(N) )

delta error (i) = error (i) at t+1 - error (i) at t

cluster mistake = | correct output - cluster output |

This model says that for positive constants A and B:

delta error = A * ( cluster mistake - B )

This equation has the property that the error increase or decrease is proportional to the size of the cluster mistake. The equilibrium is when the mistake equals B. An assumption is made that an individual net's mistake is a guassian random variable $z_i$ with mean and variance equal to its error. For the purposes of this analysis, the judge uses a convex combination of the net outputs to form the cluster output. Using the assumptions of this model, it can be shown that a strategy of increasing the relative weight in the convex combination of a net that has a relatively small error and conversely decreasing the relative weight for poorly performing nets. (1,2) is an example weight adjustment rule. This rule has the effect of increasing the weight of a network that produced a network deviation that was smaller than average. The opposite effect is seen for a network that produced a network deviation that was larger than average.

### 6.1.1     References.

D.E. Rumelhart, J.L. McClelland, and the PDP Research Group. Parallel Distributed Processing (PDP): Exploration in the Microstructure of Cognition (Vol. 1). MIT Press, Cambridge, Massachusetts, 1986.

R.P. Lippman. An Introduction to Computing with Neural Nets. IEEE ASSP magazine, Vol. 4, pp. 4-22, April, 1987.

F.F. Soulie, P. Gallinari, Y. Le Cun, and S. Thiria. Evaluation of network architectures on test learning tasks. IEEE First International Conference on Neural Networks, San Diego, pp. II653-II660, June 1987.

J. Bernasconi. Analysis and Comparison of Different Learning Algorithms for Pattern Association Problems. Neural Information Processing Systems, Denver, Co, pp. 72-81, 1987.

Abraham Lincoln. Personal communication.

## Footnotes

* also with Hughes Aircraft Company

† to whom the correspondence should be addressed
